# Evaluating the inverse decision-making approach to preference learning

**Alan Jern**
Department of Psychology
Carnegie Mellon University
ajern@cmu.edu

**Christopher G. Lucas**
Department of Psychology
Carnegie Mellon University
cglucas@andrew.cmu.edu

**Charles Kemp**
Department of Psychology
Carnegie Mellon University
ckemp@cmu.edu

## Abstract

Psychologists have recently begun to develop computational accounts of how people infer others' preferences from their behavior. The inverse decision-making approach proposes that people infer preferences by inverting a generative model of decision-making. Existing data sets, however, do not provide sufficient resolution to thoroughly evaluate this approach. We introduce a new preference learning task that provides a benchmark for evaluating computational accounts and use it to compare the inverse decision-making approach to a feature-based approach, which relies on a discriminative combination of decision features. Our data support the inverse decision-making approach to preference learning.

A basic principle of decision-making is that knowing people's preferences allows us to predict how they will behave: if you know your friend likes comedies and hates horror films, you can probably guess which of these options she will choose when she goes to the theater. Often, however, we do not know what other people like and we can only infer their preferences from their behavior. If you know that a different friend saw a comedy today, does that mean that he likes comedies in general? The conclusion you draw will likely depend on what else was playing and what movie choices he has made in the past.

A goal for social cognition research is to develop a computational account of people's ability to infer others' preferences. One computational approach is based on inverse decision-making. This approach begins with a model of how someone's preferences lead to a decision. Then, this model is inverted to determine the most likely preferences that motivated an observed decision. An alternative approach might simply learn a functional mapping between features of an observed decision and the preferences that motivated it. For instance, in your friend's decision to see a comedy, perhaps the more movie options he turned down, the more likely it is that he has a true preference for comedies. The difference between the inverse decision-making approach and the feature-based approach maps onto the standard dichotomy between generative and discriminative models.

Economists have developed an instance of the inverse decision-making approach known as the multinomial logit model [1] that has been widely used to infer consumer's preferences from their choices. This model has recently been explored as a psychological model [2, 3, 4], but there are few behavioral data sets for evaluating it as a model of how people learn others' preferences. Additionally, the data sets that do exist tend to be drawn from the developmental literature, which focuses on simple tasks that collect only one or two judgments from children [5, 6, 7]. The limitations of these data sets make it difficult to evaluate the multinomial logit model with respect to alternative accounts of preference learning like the feature-based approach.

In this paper, we use data from a new experimental task that elicits a detailed set of preference judgments from a single participant in order to evaluate the predictions of several preference learning models from both the inverse decision-making and feature-based classes. Our task requires each participant to sort a large number of observed decisions on the basis of how strongly they indicate

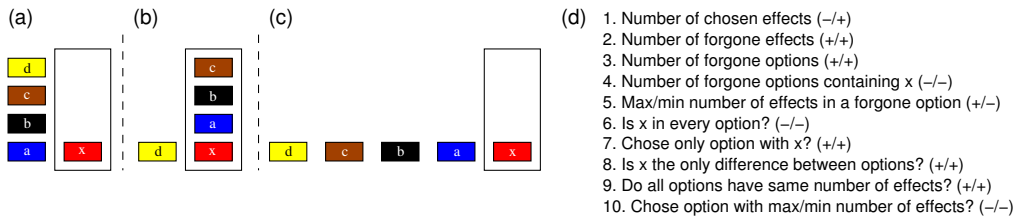

(a)     (b)     (c)     (d) 1. Number of chosen effects (–/+)
2. Number of forgone effects (+/+)
3. Number of forgone options (+/+)
4. Number of forgone options containing x (–/–)
5. Max/min number of effects in a forgone option (+/–)
6. Is x in every option? (–/–)
7. Chose only option with x? (+/+)
8. Is x the only difference between options? (+/+)
9. Do all options have same number of effects? (+/+)
10. Chose option with max/min number of effects? (–/–)

Figure 1: (a)–(c) Examples of the decisions used in the experiments. Each column represents one option and the boxes represent different effects. The chosen option is indicated by the black rectangle. (d) Features used by the weighted feature and ranked feature models. Features 5 and 10 involved maxima in Experiment 1, which focused on all positive effects, and minima in Experiment 2, which focused on all negative effects. The signs in parentheses indicate the direction of the feature that suggests a stronger preference in Experiment 1 / Experiment 2.

a preference for a chosen item. Because the number of decisions is large and these decisions vary on multiple dimensions, predicting how people will order them offers a challenging benchmark on which to compare computational models of preference learning. Data sets from these sorts of detailed tasks have proved fruitful in other domains. For example, data reported by Shepard, Hovland, and Jenkins [8]; Osherson, Smith, Wilkie, López, and Shafir [9]; and Wasserman, Elek, Chatlosh, and Baker [10] have motivated much subsequent research on category learning, inductive reasoning, and causal reasoning, respectively.

We first describe our preference learning task in detail. We then present several inverse decision-making and feature-based models of preference learning and compare these models' predictions to people's judgments in two experiments. The data are well predicted by models that follow the inverse decision-making approach, suggesting that this computational approach may help explain how people learn others' preferences.

# 1   Multi-attribute decisions and revealed preferences

We designed a task that can be used to elicit a large number of preference judgments from a single participant. The task involves a set of observed multi-attribute decisions, some examples of which are represented visually in Figure 1. Each decision is among a set of options and each option produces a set of effects. Figure 1 shows several decisions involving a total of five effects distributed among up to five options. The differently colored boxes represent different effects and the chosen option is marked by a black rectangle. For example, 1a shows a choice between an option with four effects and an option with a single effect; here, the decision maker chose the second option.

In our task, people are asked to rank a large number of these decisions by how strongly they suggest that the decision maker had a preference for a particular effect (e.g., effect x in Figure 1). By imposing some minimal constraints, the space of unique multi-attribute decisions is finite and we can obtain rankings for every decision in the space. For example, Figure 2c shows a complete list of 47 unique decisions involving up to five effects, subject to several constraints described later.

Three of these decisions are shown in Figure 1. If all the effects are positive—pieces of candy, for example—the first decision (1a) suggests a strong preference for candy x, because the decision maker turned down four pieces in favor of one. The second decision (1b), however, offers much weaker evidence because nearly everyone would choose four pieces of candy over one, even without a specific preference for x. The third decision (1c) provides evidence that is strong but perhaps not quite as strong as the first decision. When all effects are negative—like electric shocks at different body locations—decision makers may still find some effects more tolerable than others, but different inferences are sometimes supported. For example, for negative effects, 1a provides weak evidence that x is relatively tolerable because nearly everyone would choose one shock over four.

# 2   A computational account of preference learning

We now describe a simple computational model for learning a person's preferences after observing that person make a decision like the ones in Figure 1. We assume that there are $n$ available options

$\{o_1, \ldots, o_n\}$, each of which produces one or more effects from the set $\{f_1, f_2, ..., f_m\}$. For simplicity, we assume that effects are binary. Let $u_i$ denote the utility the decision maker assigns to effect $f_i$.

We begin by specifying a model of decision-making that makes the standard assumptions that decision makers tend to choose things with greater utility and that utilities are additive. That is, if $\mathbf{f_j}$ is a binary vector indicating the effects produced by option $o_j$ and $\mathbf{u}$ is a vector of utilities assigned to each of the $m$ effects, then the total utility associated with option $o_j$ can be expressed as $U_j = \mathbf{f_j}^T \mathbf{u}$. We complete the specification of the model by applying the Luce choice rule [11], a common psychological model of choice behavior, as the function that chooses among the options:

$$p(c = o_j | \mathbf{u}, \mathbf{f}) = \frac{\exp(U_j)}{\sum_{k=1}^{n} \exp(U_k)} = \frac{\exp(\mathbf{f_j}^T \mathbf{u})}{\sum_{k=1}^{n} \exp(\mathbf{f_k}^T \mathbf{u})} \tag{1}$$

where $c$ denotes the choice made.

This model can predict the choice someone will make among a specified set of options, given the utilities that person assigns to the effects in each option. To obtain estimates of someone's utilities, we invert this model by applying Bayes' rule:

$$p(\mathbf{u} | c, \mathbf{F}) = \frac{p(c | \mathbf{u}, \mathbf{F}) p(\mathbf{u})}{p(c | \mathbf{F})} \tag{2}$$

where $\mathbf{F} = \{\mathbf{f_1}, \ldots, \mathbf{f_n}\}$ specifies the available options and their corresponding effects. This is the multinomial logit model [1], a standard econometric model. In order to apply Equation 2 we must specify a prior $p(\mathbf{u})$ on the utilities. We adopt a standard approach that places independent Gaussian priors on the utilities: $u_i \sim \mathcal{N}(\mu, \sigma^2)$. For decisions where effects are positive—like candies—we set $\mu = 2\sigma$, which corresponds to a prior distribution that places approximately 2% of the probability mass below zero. Similarly, for negative effects—like electric shocks—we set $\mu = -2\sigma$.

## 2.1 Ordering a set of observed decisions

Equation 2 specifies a posterior probability distribution over utilities for a single observed decision but does not provide a way to compare the inferences drawn from multiple decisions for the purposes of ordering them. Suppose we are interested in a decision maker's preference for effect x and we wish to order a set of decisions by how strongly they support this preference. Two criteria for ordering the decisions are as follows:

Absolute utility $\qquad E(u_x | c, \mathbf{F}) = E_{u_x} \left( \frac{p(c | u_x, \mathbf{F}) p(u_x)}{p(c | \mathbf{F})} \right)$

Relative utility $\qquad p(\forall j\ u_x \geq u_j | c, \mathbf{F}) = \frac{p(c | \forall j\ u_x \geq u_j, \mathbf{F}) p(\forall j\ u_x \geq u_j)}{p(c | \mathbf{F})}$

The absolute utility model orders decisions by the mean posterior utility for effect x. This criterion is perhaps the most natural way to assess how much a decision indicates a preference for x, but it requires an inference about the utility of x in isolation, and research suggests that people often think about the utility of an effect only in relation to other salient possibilities [12]. The relative utility model applies this idea to preference learning by ordering decisions based on how strongly they suggest that x has a greater utility than all other effects. The decisions in Figures 1b and 1c are cases where the two models lead to different predictions. If the effects are all negative (e.g., electric shocks), the absolute utility model predicts that 1b provides stronger evidence for a tolerance for x because the decision maker chose to receive four shocks instead of just one. The relative utility model predicts that 1c provides stronger evidence because 1b offers no way to determine the relative tolerance of the four chosen effects with respect to one another.

Like all generative models, the absolute and relative models incorporate three qualitatively different components: the likelihood term $p(c | \mathbf{u}, \mathbf{F})$, the prior $p(\mathbf{u})$, and the reciprocal of the marginal likelihood $1/p(c | \mathbf{F})$. We assume that the total number of effects is fixed in advance and, as a result, the prior term will be the same for all decisions that we consider. The two other components, however, will vary across decisions. The inverse decision-making approach predicts that both components should influence preference judgments, and we will test this prediction by comparing our

two inverse decision-making models to two alternatives that rely only one of these components as an ordering criterion:

| | |
|---|---|
| Representativeness | $p(c|\forall j\ u_x \geq u_j, \mathbf{F})$ |
| Surprise | $1/p(c|\mathbf{F})$ |

The representativeness model captures how likely the observed decision would be if the utility for x were high, and previous research has shown that people sometimes rely on a representativeness computation of this kind [13]. The surprise model captures how unexpected the observed decision is overall; surprising decisions may be best explained in terms of a strong preference for x, but unsurprising decisions provide little information about x in particular.

## 2.2 Feature-based models

We also consider a class of feature-based models that use surface features to order decisions. The ten features that we consider are shown in Figure 1d, where x is the effect of interest. As an example, the first feature specifies the number of effects chosen; because x is always among the chosen effects, decisions where few or no other effects belong to the chosen option suggest the strongest preference for x (when all effects are positive). This and the second feature were previously identified by Newtson [14]; we included the eight additional features shown in Figure 1d in an attempt to include all possible features that seemed both simple and relevant.

We consider two methods for combining this set of features to order a set of decisions by how strongly they suggest a preference for x. The first model is a standard linear regression model, which we refer to as the *weighted feature* model. The model learns a weight for each feature, and the rank of a given decision is determined by a weighted sum of its features. The second model is a *ranked feature* model that sorts the observed decisions with respect to a strict ranking of the features. The top-ranked feature corresponds to the primary sort key, the second-ranked feature to the secondary sort key, and so on. For example, suppose that the top-ranked feature is the number of chosen effects and the second-ranked feature is the number of forgone options. Sorting the three decisions in Figure 1 according to this criterion produces the following ordering: 1a,1c,1b. This notion of sorting items on the basis of ranked features has been applied before to decision-making [15, 16] and other domains of psychology [17], but we are not aware of any previous applications to preference learning.

Although our inverse decision-making and feature-based models represent two very different approaches, both may turn out to be valuable. An inverse decision-making approach may be the appropriate account of preference learning at Marr's [18] computational level, and a feature-based approach may capture the psychological processes by which the computational-level account is implemented. Our goal, therefore, is not necessarily to accept one of these approaches and dismiss the other. Instead, we entertain three distinct possibilities. First, both approaches may account well for the data, which would support the idea that they are valid accounts operating at different levels of analysis. Second, the inverse decision-making approach may offer a better account, suggesting that process-level accounts other than the feature-based approach should be explored. Finally, the feature-based approach may offer a better account, suggesting that inverse decision-making does not constitute an appropriate computational-level account of preference learning.

## 3 Experiment 1: Positive effects

Our first experiment focuses on decisions involving only positive effects. The full set of 47 decisions we used is shown in Figure 2c. This set includes every possible unique decision with up to five different effects, subject to the following constraints: (1) one of the effects (effect x) must always appear in the chosen option, (2) there are no repeated options, (3) each effect may appear in an option at most once, (4) only effects in the chosen option may be repeated in other options, and (5) when effects appear in multiple options, the number of effects is held constant across options. The first constraint is necessary for the sorting task, the second two constraints create a finite space of decisions, and the final two constraints limit attention to what we deemed the most interesting cases.

**Method** 43 Carnegie Mellon undergraduates participated for course credit. Each participant was given a set of cards, with one decision printed on each card. The decisions were represented visually

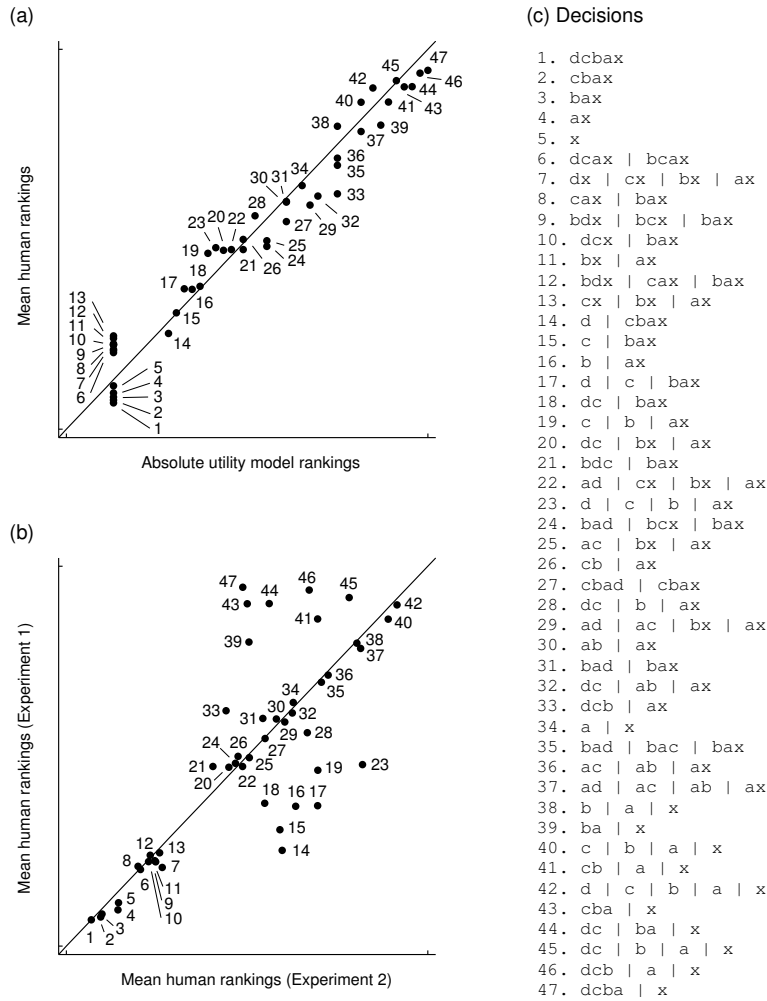

(a)

Mean human rankings

Absolute utility model rankings

(b)

Mean human rankings (Experiment 1)

Mean human rankings (Experiment 2)

(c) Decisions

```
1.  dcbax
2.  cbax
3.  bax
4.  ax
5.  x
6.  dcax | bcax
7.  dx | cx | bx | ax
8.  cax | bax
9.  bdx | bcx | bax
10. dcx | bax
11. bx | ax
12. bdx | cax | bax
13. cx | bx | ax
14. d | cbax
15. c | bax
16. b | ax
17. d | c | bax
18. dc | bax
19. c | b | ax
20. dc | bx | ax
21. bdc | bax
22. ad | cx | bx | ax
23. d | c | b | ax
24. bad | bcx | bax
25. ac | bx | ax
26. cb | ax
27. cbad | cbax
28. dc | b | ax
29. ad | ac | bx | ax
30. ab | ax
31. bad | bax
32. dc | ab | ax
33. dcb | ax
34. a | x
35. bad | bac | bax
36. ac | ab | ax
37. ad | ac | ab | ax
38. b | a | x
39. ba | x
40. c | b | a | x
41. cb | a | x
42. d | c | b | a | x
43. cba | x
44. dc | ba | x
45. dc | b | a | x
46. dcb | a | x
47. dcba | x
```

Figure 2: (a) Comparison between the absolute utility model rankings and the mean human rankings for Experiment 1. Each point represents one decision, numbered with respect to the list in panel c. (b) Comparison between the mean human rankings in Experiments 1 and 2. In both scatter plots, the solid diagonal lines indicate a perfect correspondence between the two sets of rankings. (c) The complete set of decisions, ordered by the mean human rankings from Experiment 1. Options are separated by vertical bars and the chosen option is always at the far right. Participants were always asked about a preference for effect x.

as in Figure 1 but without the letter labels. Participants were told that the effects were different types of candy and each option was a bag containing one or more pieces of candy. They were asked to sort the cards by how strongly each decision suggested that the decision maker liked a particular target candy, labeled x in Figure 2c. They sorted the cards freely on a table but reported their final rankings by writing them on a sheet of paper, from weakest to strongest evidence. They were instructed to order the cards as completely as possible, but were told that they could assign the same ranking to a set of cards if they believed those cards provided equal evidence.

### 3.1 Results

Two participants were excluded as outliers based on the criterion that their rankings for at least five decisions were at least three standard deviations from the mean rankings. We performed a hierarchical clustering analysis of the remaining 41 participants' rankings using rank correlation as a similarity metric. Participants' rankings were highly correlated: cutting the resulting dendrogram at 0.2 resulted in one cluster that included 33 participants and the second largest cluster included

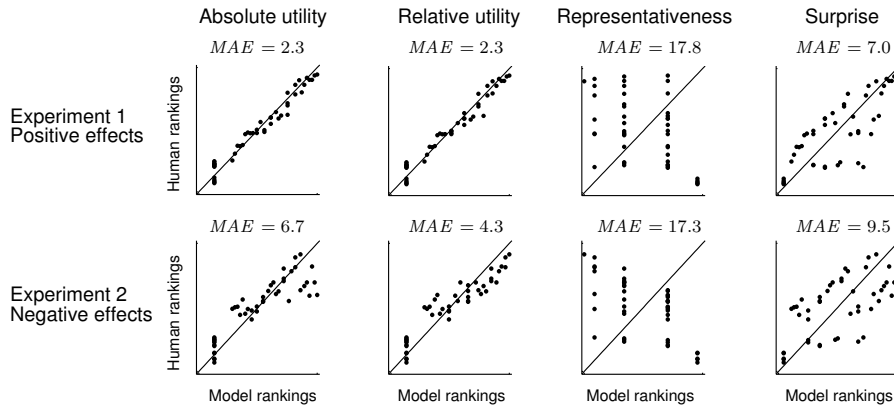

Figure 3: Comparison between human rankings in both experiments and predicted rankings from four models. The solid diagonal lines indicate a perfect correspondence between human and model rankings.

only 3 participants. Thus, we grouped all participants together and analyzed their mean rankings. The 0.2 threshold was chosen because it produced the most informative clustering in Experiment 2.

**Inverse decision-making models**   We implemented the inverse decision-making models using importance sampling with 5 million samples drawn from the prior distribution $p(\mathbf{u})$. Because all the effects were positive, we used a prior on utilities that placed nearly all probability mass above zero ($\mu = 4$, $\sigma = 2$).

The mean human rankings are compared with the absolute utility model rankings in Figure 2a, and the mean human rankings are listed in order in 2c. Fractional rankings were used for both the human data and the model predictions. The human rankings in the figure are the means of participants' fractional rankings. The first row of Figure 3 contains similar plots that allow comparison of the four models we considered. In these plots, the solid diagonal lines indicate a perfect correspondence between model and human rankings. Thus, the largest deviations from this line represent the largest deviations in the data from the model's predictions.

Figure 3 shows that the absolute and relative utility models make virtually identical predictions and both models provide a strong account of the human rankings as measured by mean absolute error ($MAE = 2.3$ in both cases). Moreover, both models correctly predict the highest ranked decision and the set of lowest ranked decisions. The only clear discrepancy between the model predictions and the data is the cluster of points at the lower left, labeled as Decisions 6–13 in Figure 2a. These are all cases in which effect x appears in all options and therefore these decisions provide no information about a decision maker's preference for x. Consequently, the models assign the same ranking to this group as to the group of decisions in which there is only a single option (Decisions 1–5). Although people appeared to treat these groups somewhat differently, the models still correctly predict that the entire group of decisions 1–13 is ranked lower than all other decisions.

The surprise and representativeness models do not perform nearly as well ($MAE = 7.0$ and $17.8$, respectively). Although the surprise model captures some of the general trends in the human rankings, it makes several major errors. For example, consider Decision 7: `dx|cx|bx|ax`. This decision provides no information about a preference for x because it appears in every option. The decision is surprising, however, because a decision maker choosing at random from these options would make the observed choice only $1/4$ of the time. The representativeness model performs even worse, primarily because it does not take into account alternative explanations for why an option was chosen, such as the fact that no other options were available (e.g., Decision 1 in Figure 2c). The failure of these models to adequately account for the data suggests that both the likelihood $p(c|\mathbf{u}, \mathbf{F})$ and marginal likelihood $p(c|\mathbf{F})$ are important components of the absolute and relative utility models.

**Feature-based models**   We compared the performance of the absolute and relative utility models to our two feature-based models: the weighted feature and ranked feature models. For each participant,

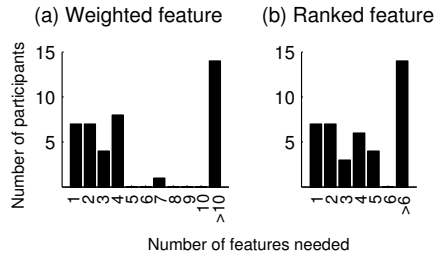

Figure 4: Results of the feature-based model analysis from Experiment 1 for (a) the weighted feature models and (b) the ranked feature models. The histograms show the minimum number of features needed to match the accuracy (measured by *MAE*) of the absolute utility model for each participant.

we considered every subset of features[1] in Figure 1d in order to determine the minimum number of features needed by the two models to achieve the same level of accuracy as the absolute utility model, as measured by mean absolute error. The results of these analyses are shown in Figure 4. For the majority of participants, at least four features were needed by both models to match the accuracy of the absolute utility model. For the weighted feature model, 14 participants could not be fit as well as the absolute utility model even when all ten features were considered. These results indicate that a feature-based account of people's inferences in our task must be supplied with a relatively large number of features. By contrast, the inverse decision-making approach provides a relatively parsimonious account of the data.

## 4   Experiment 2: Negative effects

Experiment 2 focused on a setting in which all effects are negative, motivated by the fact that the inverse decision-making models predict several major differences in orderings when effects are negative rather than positive. For instance, the absolute utility model's relative rankings of the decisions in Figures 1a and 1b are reversed when all effects are negative rather than positive.

**Method**   42 Carnegie Mellon undergraduates participated for course credit. The experimental design was identical to Experiment 1 except that participants were told that the effects were electric shocks at different body locations. They were asked to sort the cards on the basis of how strongly each decision suggested that the decision maker finds shocks at the target location relatively tolerable. The model predictions were derived in the same way as for Experiment 1, but with a prior distribution on utilities that placed nearly all probability mass below zero ($\mu = -4$, $\sigma = 2$) to reflect the fact that effects were all negative.

### 4.1   Results

Three participants were excluded as outliers by the same criterion applied in Experiment 1. The resulting mean rankings are compared with the corresponding rankings from Experiment 1 in Figure 2b. The figure shows that responses based on positive and negative effects were substantially different in a number of cases. Figure 3 shows how the mean rankings compare to the predictions of the four models we considered. Although the relative utility model is fairly accurate, no model achieves the same level of accuracy as the absolute and relative utility models in Experiment 1. In addition, the relative utility model provides a poor account of the responses of many individual participants. To better understand responses at the individual level, we repeated the hierarchical clustering analysis described in Experiment 1, which revealed that 29 participants could be grouped into one of four clusters, with the remaining participants each in their own clusters. We analyzed these four clusters independently, excluding the 10 participants that could not be naturally grouped.

We compared the mean rankings of each cluster to the absolute and relative utility models, as well as all one- and two-feature weighted feature and ranked feature models. Figure 5 shows that the mean rankings of participants in Cluster 1 ($N = 8$) were best fit by the absolute utility model, the mean rankings of participants in Cluster 2 ($N = 12$) were best fit by the relative utility model, and the mean rankings of participants in Clusters 3 ($N = 3$) and 4 ($N = 6$) were better fit by feature-based models than by either the absolute or relative utility models.

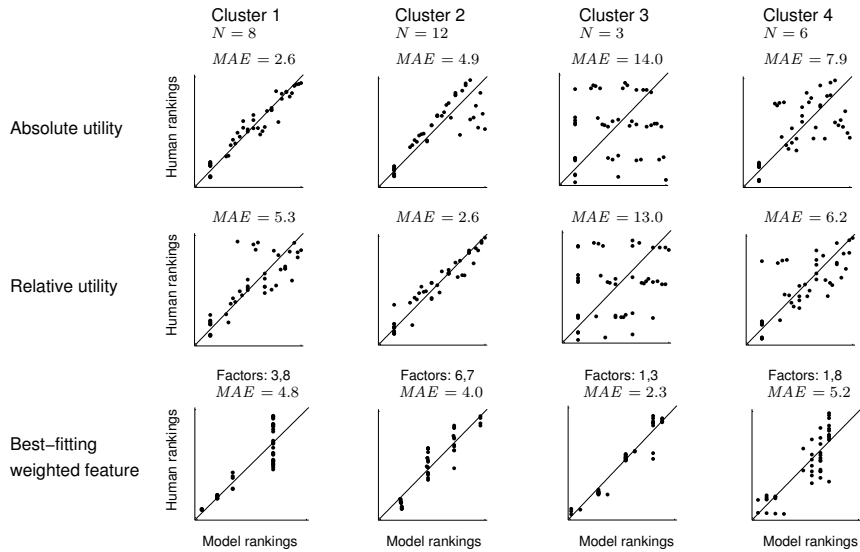

Figure 5: Comparison between human rankings for four clusters of participants identified in Experiment 2 and predicted rankings from three models. Each point in the plots corresponds to one decision and the solid diagonal lines indicate a perfect correspondence between human and model rankings. The third row shows the predictions of the best-fitting two-factor weighted feature model for each cluster. The two factors listed refer to Figure 1d.

To examine how well the models accounted for individuals' rankings within each cluster, we compared the predictions of the inverse decision-making models to the best-fitting two-factor feature-based model for each participant. In Cluster 1, 7 out of 8 participants were best fit by the absolute utility model; in Cluster 2, 8 out of 12 participants were best fit by the relative utility model; in Clusters 3 and 4, all participants were better fit by feature-based models. No single feature-based model provided the best fit for more than two participants, suggesting that participants not fit well by the inverse decision-making models were not using a single alternative strategy.

Applying the feature-based model analysis from Experiment 1 to the current results revealed that the weighted feature model required an average of 6.0 features to match the performance of the absolute utility model for participants in Cluster 1, and an average of 3.9 features to match the performance of the relative utility model for participants in Cluster 2. Thus, although a single model did not fit all participants well in the current experiment, many participants were fit well by one of the two inverse decision-making models, suggesting that this general approach is useful for explaining how people reason about negative effects as well as positive effects.

## 5   Conclusion

In two experiments, we found that an inverse decision-making approach offered a good computational account of how people make judgments about others' preferences. Although this approach is conceptually simple, our analyses indicated that it captures the influence of a fairly large number of relevant decision features. Indeed, the feature-based models that we considered as potential process models of preference learning could only match the performance of the inverse decision-making approach when supplied with a relatively large number of features. We feel that this result rules out the feature-based approach as psychologically implausible, meaning that alternative process-level accounts will need to be explored. One possibility is sampling, which has been proposed as a psychological mechanism for approximating probabilistic inferences [19, 20]. However, even if process models that use large numbers of features are considered plausible, the inverse decision-making approach provides a valuable computational-level account that helps to explain which decision features are informative.

**Acknowledgments**   This work was supported in part by the Pittsburgh Life Sciences Greenhouse Opportunity Fund and by NSF grant CDI-0835797.

## Footnotes

[1]A maximum of six features was considered for the ranked feature model because considering more features was computationally intractable.

# References

[1] D. McFadden. Conditional logit analysis of qualitative choice behavior. In P. Zarembka, editor, *Frontiers in Econometrics*. Amademic Press, New York, 1973.

[2] C. G. Lucas, T. L. Griffiths, F. Xu, and C. Fawcett. A rational model of preference learning and choice prediction by children. In *Proceedings of Neural Information Processing Systems 21*, 2009.

[3] L. Bergen, O. R. Evans, and J. B. Tenenbaum. Learning structured preferences. In *Proceedings of the 32nd Annual Conference of the Cognitive Science Society*, 2010.

[4] A. Jern and C. Kemp. Decision factors that support preference learning. In *Proceedings of the 33rd Annual Conference of the Cognitive Science Society*, 2011.

[5] T. Kushnir, F. Xu, and H. M. Wellman. Young children use statistical sampling to infer the preferences of other people. *Psychological Science*, 21(8):1134–1140, 2010.

[6] L. Ma and F. Xu. Young children's use of statistical sampling evidence to infer the subjectivity of preferences. *Cognition*, in press.

[7] M. J. Doherty. *Theory of Mind: How Children Understand Others' Thoughts and Feelings*. Psychology Press, New York, 2009.

[8] R. N. Shepard, C. I. Hovland, and H. M. Jenkins. Learning and memorization of classifications. *Psychological Monographs*, 75, Whole No. 517, 1961.

[9] D. N. Osherson, E. E. Smith, O. Wilkie, A. López, and E. Shafir. Category-based induction. *Psychological Review*, 97(2):185–200, 1990.

[10] E. A. Wasserman, S. M. Elek, D. L. Chatlosh, and A. G. Baker. Rating causal relations: Role of probability in judgments of response-outcome contingency. *Journal of Experimental Psychology: Learning, Memory, and Cognition*, 19(1):174–188, 1993.

[11] R. D. Luce. *Individual choice behavior*. John Wiley, 1959.

[12] D. Ariely, G. Loewenstein, and D. Prelec. Tom Sawyer and the construction of value. *Journal of Economic Behavior & Organization*, 60:1–10, 2006.

[13] D. Kahneman and A. Tversky. Subjective probability: A judgment of representativeness. *Cognitive Psychology*, 3(3):430–454, 1972.

[14] D. Newtson. Dispositional inference from effects of actions: Effects chosen and effects forgone. *Journal of Experimental Social Psychology*, 10:489–496, 1974.

[15] P. C. Fishburn. Lexicographic orders, utilities and decision rules: A survey. *Management Science*, 20(11):1442–1471, 1974.

[16] G. Gigerenzer and P. M. Todd. *Fast and frugal heuristics: The adaptive toolbox*. Oxford University Press, New York, 1999.

[17] A. Prince and P. Smolensky. *Optimality Theory: Constraint Interaction in Generative Grammar*. Wiley-Blackwell, 2004.

[18] D. Marr. *Vision*. W. H. Freeman, San Francisco, 1982.

[19] A. N. Sanborn, T. L. Griffiths, and D. J. Navarro. Rational approximations to rational models: Alternative algorithms for category learning. *Psychological Review*, 117:1144–1167, 2010.

[20] L. Shi and T. L. Griffiths. Neural implementation of Bayesian inference by importance sampling. In *Proceedings of Neural Information Processing Systems 22*, 2009.

